# Deep Alternative Neural Network: Exploring Contexts as Early as Possible for Action Recognition

**Jinzhuo Wang, Wenmin Wang, Xiongtao Chen, Ronggang Wang, Wen Gao[†]**
School of Electronics and Computer Engineering, Peking University
[†]School of Electronics Engineering and Computer Science, Peking University
jzwang@pku.edu.cn, wangwm@ece.pku.edu.cn
cxt@pku.edu.cn, rgwang@ece.pku.edu.cn, wgao@pku.edu.cn

## Abstract

Contexts are crucial for action recognition in video. Current methods often mine contexts after extracting hierarchical local features and focus on their high-order encodings. This paper instead explores contexts as early as possible and leverages their evolutions for action recognition. In particular, we introduce a novel architecture called deep alternative neural network (DANN) stacking alternative layers. Each alternative layer consists of a volumetric convolutional layer followed by a recurrent layer. The former acts as local feature learner while the latter is used to collect contexts. Compared with feed-forward neural networks, DANN learns contexts of local features from the very beginning. This setting helps to preserve hierarchical context evolutions which we show are essential to recognize similar actions. Besides, we present an adaptive method to determine the temporal size for network input based on optical flow energy, and develop a volumetric pyramid pooling layer to deal with input clips of arbitrary sizes. We demonstrate the advantages of DANN on two benchmarks HMDB51 and UCF101 and report competitive or superior results to the state-of-the-art.

## 1 Introduction

Contexts contribute semantic clues for action recognition in video. Current leading convolutional neural networks (CNNs) [13, 22, 31] and its shifted version 3D CNNs [11, 28, 29] often aggregate contexts in the *late stage*. More precisely, in the first layer of a typical CNN, receptive field (RF) starts at the kernel size which is usually small and the outputs only extract local features. As the layer goes deeper, RF expands and contexts start to be involved. These models need to be very deep [32] to preserve rich context topologies and reach competitive trajectory-based works [16, 19, 20, 30]. We speculate this is the main reason that going deeper with convolutions achieves better performance on many visual recognition tasks [23, 26]. However, it is not wise to simply increase layer number due to parameter burden. Besides, these models do not embed context evolutions of local features in the forward flow which is essential for context mining [17, 18]. To this end, we attempt to explore contexts *as early as possible* and investigate architectures for action recognition.

Our motivation also derives from the relations between CNNs and visual systems of brain since they share many properties [9, 10]. One remarkable difference is that abundant recurrent connections exist in the visual system of brain [3] while CNNs only have forward connections. Anatomical evidences have shown that recurrent synapses typically outnumber feed-forward, top-down and feedback synapses in the neocortex [4, 37]. This makes visual recognition tend to be a dynamic procedure. Hence, we investigate to insert recurrent connections in the deployment of our architecture. Recent works utilize recurrent neural networks (RNNs) with long-short term memory (LSTM) units at the end CNN-based features of each frame to exploit semantic combinations [5, 25, 35]. These

methods can be regarded as inter-image context learner. In contrast, we attempt to apply recurrent connections to each level of hierarchical features to aggregate their context evolutions. Similar efforts have demonstrated effectivity for image analysis such as object recognition and scene parsing [21, 17, 18]. We extend in temporal domain and study its potential for action recognition.

The main contribution of this paper is summarized as follows. First, we propose a deep alternative neural network (DANN) for action recognition. DANN stacks alternative layers consisting of volumetric convolutional layer and recurrent layer. The alternative deployment is used to preserve the contexts of local features as early as possible and embed their evolutions in the hierarchical feature learning procedure. Second, we introduce an adaptive method to determine the temporal size of input video clip based on the density of optical flow energy. Instead of manual choices used in most deep architectures, our method utilizes adaptive input video clips preserving long range dependencies, while not breaking semantic structures. To cope with input video clips of arbitrary sizes, we develop a volumetric pyramid pooling layer to resize the output to fixed-size before fully connected layers. Finally, we conduct extensive experiments and demonstrate the benefits of our method with *early context exploration*. On two challenging benchmarks HMDB51 and UCF101, we report competitive or superior results to the state-of-the-art.

## 2 Deep Alternative Neural Network

### 2.1 Adaptive Network Input

The input size of deep networks in temporal domain is often determined empirically since it is hard to evaluate all the choices. Previous methods often consider short intervals such as [11, 13, 27, 28] from 1 to 16 frames. Recent work [29] argues that human actions usually span tens or hundreds of frames and contain characteristic patterns with long-term temporal structure. The authors use 60-frame as the network input and demonstrate its advantage over 16-frame. However, it is still an ad hoc manner and difficult to favor all the action classes. We introduce an adaptive method to automatically select the most discriminative video fragments using the density of optical flow energy. We attempt to preserve as much as motion information and appropriate range dependencies while not breaking their semantic structures in temporal domain.

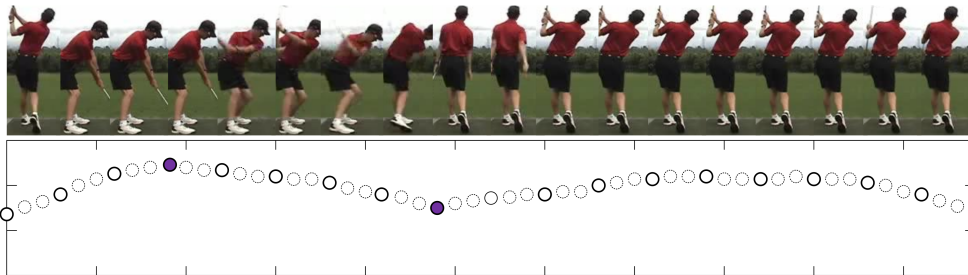

Figure 1: Sample video clip at every 3 frames of class "golfswing" and its optical flow energy with local minima and maxima landmarks where landmarks approximately correspond to motion change.

Many evidences show that motion energy intensity induced by human activity exhibits regular periodicity [33]. This signal can be approximately estimated by optical flow computation as shown in Figure 1, and is particularly suitable to address our temporal estimation due to: (1) the local minima and maxima landmarks probably correspond to characteristic gesture and motion; (2) it is relatively robust to changes in camera viewpoint. More specifically, we first compute the optical flow field $(\mathbf{v}_x, \mathbf{v}_y)$ for each frame $I$ from a video $Q$ and define its flow energy as

$$e(I) = \sum_{(x,y) \in P} \|\mathbf{v}_x(x,y), \mathbf{v}_y(x,y)\|_2 \qquad (1)$$

where $P$ is the pixel level set of selected interest points. The energy of $Q$ is then obtained as $\mathcal{E} = \{e(I_1), \cdots, e(I_t)\}$, which is further smoothed by a Gaussian filter to suppress noise. Subsequently, we locate the local minima and maxima landmarks $\{t\}$ of $\mathcal{E}$ and for each two consecutive landmarks create a video fragment $\mathbf{s}$ by extracting the frames $\mathbf{s} = \{I_{t-1}, \cdots, I_t\}$. We average the fragment

length of each class and illustrate the distribution in Figure 2, which indicates that using a universal length can not favor all classes. To deal with the different length of video clip, we adopt the idea of spatial pyramid pooling (SPP) in [8] and extend to temporal domain, developing a volumetric pyramid pooling (VPP) layer to transfer video clip of arbitrary size into a universal length in the last alternative layer before fully connected layer, which is presented in Section 2.3.

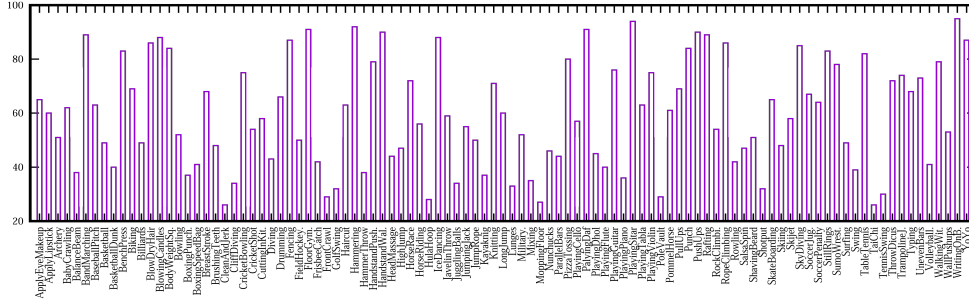

Figure 2: Average fragment length of each class in UCF101 dataset.

## 2.2 Alternative Layer

The key component of DANN is the alternative layer (AL), which consists of a standard volumetric convolutional layer followed by a designed recurrent layer. Specifically, volumetric convolution is first performed to extract features from local spatiotemporal neighborhoods on feature maps in the previous layers. Then a recurrent layer is applied to the output and iteratively proceeds for $T$ times. This procedure makes each unit evolve over discrete time steps and aggregate larger RFs. More formally, the input of a unit at position $(x, y, z)$ in the $j$th feature map of the $i$th AL at time $t$, denoted as $u_{ij}^{xyz}(t)$, is given by

$$
\begin{aligned}
u_{ij}^{xyz}(t) &= u_{ij}^{xyz}(0) + f(\mathbf{w}_{ij}^r u_{ij}^{xyz}(t-1)) + b_{ij} \\
u_{ij}^{xyz}(0) &= f(\mathbf{w}_{(i-1)j}^c u_{(i-1)j}^{xyz})
\end{aligned}
\tag{2}
$$

where $u_{ij}^{xyz}(0)$ denotes the feed-forward output of volumetric convolutional layer, $u_{ij}^{xyz}(t-1)$ is the recurrent input of previous time, $\mathbf{w}_k^c$ and $\mathbf{w}_k^r$ are the vectorized feed-forward kernels and recurrent kernels, $b_{ij}$ is the bias for $j$th feature map in $i$th layer, $f$ is defined as popular rectified linear unit (ReLU) function followed by a local response normalization (LRN) [14]. The first term is the output of volumetric convolution of previous layer and the second term is induced by the recurrent connections. LRN mimics the lateral inhibition in the cortex where different features compete for large responses.

Equation 2 describes the dynamic behavior of AL where contexts are involved after local features are extracted. Unfolding recurrent connection for $T$ time steps results in a feed-forward subnetwork of depth $T + 1$ as shown in Figure 3. While the recurrent input evolves over iterations, the feed-forward input remains the same in all iterations. When $t = 0$ only the feed forward input is present. The effective RF of an AL unit in the feature maps of the previous layer expands when the iteration number increases.

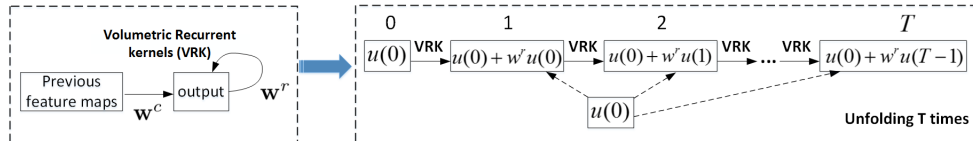

Figure 3: Illustrations of an alternative layer (left). The unfolding recurrent procedure is on the right.

The recurrent connections in AL provide two advantages. First, they enable every unit to incorporate contexts in an arbitrarily large region in the current layer. As the time steps increase, the state of every unit is influenced by other units in a larger and larger neighborhood in the current layer. As

a consequence, the size of regions that each unit can "watch" in the input space also increases. In standard volumetric convolutions, the size of effective RFs of the units in the current layer is fixed, and "watching" a larger region is only possible for units in higher layers. But unfortunately the context seen by higher-level units cannot influence the states of the units in the current layer without top-down connections. Second, the recurrent connections increase the network depth while keeping the number of adjustable parameters constant by weight sharing, since AL consumes only extra constant parameters of a recurrent kernel size compared with standard volumetric convolutional layer.

## 2.3 Volumetric Pyramid Pooling Layer

The AL accepts input video clips of arbitrary sizes and produces outputs of variable sizes. However, the fully connected layers require fixed-length vectors. Similar phenomenon can be found in region CNN (R-CNN) [6] where the input image patch is of arbitrary size. To adopt DANN for input video clips of arbitrary sizes, we replace the last pooling layer with a volumetric pyramid pooling layer (VPPL) inspired by the success of spatial pyramid pooling layer (SPPL) [8]. Figure 4 illustrates the structure of VPPL. In each volumetric bin, we pool the responses of each kernel (throughout this paper we use max pooling). The outputs of the volumetric pyramid pooling are $kM$-dimensional vectors where $M$ is the number of bins and $k$ is the number of kernels in the last alternative layer. The fixed-dimensional vectors are then sent to the fully connected layers.

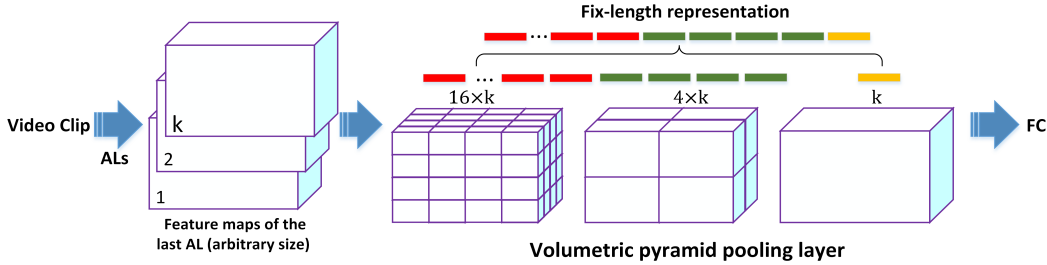

Figure 4: A network structure with volumetric pyramid pooling layer (VPPL) to resize feature maps of arbitrary size to fixed size.

With VPPL, the input video clips can be of any sizes. This allows not only arbitrary aspect ratios, but also arbitrary scales. One can apply more compact video clips only containing semantic regions such as *action tubes* in [7] to DANN with our VPPL to pursue potential improvement.

## 2.4 Overall Architecture

| AL₁ 64 | Pool₁ | AL₂ 128 | Pool₂ | AL₃ 256 | Pool₃ | AL₄ 256 | Pool₄ | AL₅ 512 | Pool₅ | AL₆ 512 | VPPL | FC₁ 2048 | FC₂ 2048 | FC₃ 2048 | Softmax |
|---|---|---|---|---|---|---|---|---|---|---|---|---|---|---|---|

Figure 5: DANN has 6 alternative layers, 5 volumetric pooling layers, 1 volumetric pyramid pooling layer, 3 fully connected layers and a softmax layer. Number of kernels are denoted in each box.

Our network architecture DANN is illustrated in Figure 5. The network has 6 alternative layers with 64, 128, 256, 256, 512 and 512 kernel response maps, followed by a volumetric pyramid pooling layer and 3 fully connected layers of size 2048 each. Following [28] we use $3 \times 3 \times 3$ kernel for volumetric convolutional layer and recurrent layers of all 6 alternative layers. After each alternative layer, the network includes a ReLU and a volumetric max pooling layer. Max pooling kernels are of size $2 \times 2 \times 2$ except in the first layer, where it is $2 \times 2 \times 1$. All of these volumetric convolutional layers and recurrent layers are applied with appropriate padding and stride in both spatial and temporal dimensions. VPPL is applied to resize the output of the last AL to fixed-size which is the input of fully connected layers. Fully connected layers are followed by ReLU layers and a softmax at the end of the network, which outputs class scores.

# 3 Implementation details

The major implementations of DANN including volumetric convolutions, recurrent layers and optimizations are derived from Torch toolbox platform [2].

**Data Augmentation.** Inspired by the random spatial cropping during training [23], we apply the corresponding augmentation to spatiotemporal dimension, which we call random clipping. During training stage, given an input video, we first determine their temporal size $t$ as discussed in Section 2.1. Then we randomly select point $(x, y, z)$ to sample a video clip of fixed size $80 \times 80 \times t$. A common alternative is to pre-process data by using a sliding window approach to have pre-segmented clips. However, this approach limits the amount of data when the windows are not overlapped as in [28]. Another data augmentation method that we evaluate is a multi-scale cropping similar to [32].

**Training.** We use SGD applied to mini-batches with negative log likelihood criterion. The size of mini-batch is set 30. Training is performed by minimizing the cross-entropy loss function using the backpropagation through time (BPTT) algorithm [34]. This is equivalent to using the standard BP algorithm on the time-unfolded network. The final gradient of a shared weight is the sum of its gradients over all time steps. The initial learning rate for networks learned from scratch is $3 \times 10^{-3}$ and it is $3 \times 10^{-4}$ for networks fine-tuned from pre-trained models. The above schedule is used together with 0.9 dropout ratio. The momentum is set to 0.9 and weight decay is initialized with $5 \times 10^{-3}$ and reduced by $10^{-1}$ factor at every decrease of the learning rate.

**Testing.** At test time, a video is also applied with temporal estimation in Section 2.1 and divided into $80 \times 80 \times t$ clips with a temporal stride of 4 frames, where $t$ is the adaptive temporal size. Each clip is further tested with 10 crops, namely 4 corners and the center, together with their horizontal flips. The video-level score is obtained by averaging all the clip-level scores and crop scores.

# 4 Evaluations

## 4.1 Datasets

The evaluation is performed on UCF101 [24] and HMDB51 [15] benchmarks. Specifically, UCF101 contains 13K videos, annotated into 101 classes while HMDB51 includes 6.8K videos of 51 actions. The evaluation protocol is the same for both datasets: the organisers provide three training and test splits, and the performance is measured by the mean classification accuracy across the splits. Each UCF101 split contains 9.5K training videos while HMDB51 split contains 3.7K training videos.

## 4.2 Quantitative Results

We first evaluate several experimental deployment choices and determine the common settings. Then we study the impact of different configurations of our DANN and investigate the optimal architecture. Finally, we report our best model and compare with state-of-the-art results.

**Optical flow quality.** We used three types of optical flow as input signal. The performance influence is summarized in Table 1(a). We observe that sparse optical flow consistently outperforms RGB. The use of TVL1 suggested in [32] allows an almost 20% increase in performance. This demonstrates that action recognition is more easy to learn from motion information compared to raw pixel values. Given such results, we choose TVL1 optical flow for all remaining experiments in this paper.

**Data augmentation.** Table 1(b) demonstrates the influence of data augmentation. Our baseline is sliding window with 75% overlap. On UCF101 split 1 dataset, we find random clipping and multi-scale clipping both outperform the baseline and their combination can further boost the performance. Thus we use the combination strategy in the following experiments.

**Temporal length.** Another issue we discuss is that our DANN takes video clips with adaptive temporal length, which is different from most existing architectures. We examine such setting by comparing 6AL_VPPL_3FC with a new architecture 6AL_3FC using fixed-size temporal length of 16-frame, 32-frame and 64-frame, while removing VPPL. The performance gain by 6AL_VPPL_3FC on UCF101 split 1 is approximate 4.2% as shown in Table 2(a). This result verifies the advantages of our adaptive method to determine temporal length for network input.

Table 1: Performance comparison of different input modalities and data augmentation strategies on UCF101 split1.

(a) Impact of optical flow quality.

| Input | Clip-level | Video-level |
|---|---|---|
| RGB | 64.4 | 64.9 |
| MPEG [12] | 71.3 | 73.5 |
| Brox [1] | 76.7 | 77.2 |
| TVL1 [36] | 78.1 | 79.6 |

(b) Impact of data augmentation using TVL1.

| Method | Clip-level | Video-level |
|---|---|---|
| Sliding window | 75.4 | 74.8 |
| Random clipping | 78.5 | 79.6 |
| Multi-scale clipping | 81.2 | 82.4 |
| Combined | 81.6 | 82.3 |

**Additional training data.** We conduct experiments to see if our spatio-temporal features learned on one dataset can help to improve the accuracy of the other one. Such additional data is already known to improve results in some gain [22]. The performance from scratch is $56.4\%$ while fine-tuning HMDB51 from UCF101 boosts the performance to $62.5\%$. Similar conclusion is demonstrated in Table 2(b). We conclude that one can learn generic representations with DANN like C3D [28].

Table 2: Performance impact of temporal length choice and additional training data.

(a) Impact of temporal length on UCF101.

| Temporal length | Clip-level | Video-level |
|---|---|---|
| 16-frame | 77.2 | 77.6 |
| 32-frame | 77.3 | 77.2 |
| 64-frame | 79.7 | 80.1 |
| Adaptive (Ours) | 82.8 | 83.0 |

(b) Impact of additional training data.

| Method | Accuracy |
|---|---|
| From scratch UCF | 80.2 |
| Fine-tuning from HMDB | 83.7 |
| From scratch HMDB | 56.4 |
| Fine-tuning from UCF | 62.5 |

**Model Analysis.** In the following we investigate the optimal configuration of our DANN. There are two crucial settings for DANN model. The first one is the AL deployment including its order and number. The other one is the unfolding time $T$ in the recurrent layers. Table 3 shows the details of performance comparison, where VC is the standard volumetric convolutional layer and B_6VC_3FC is a baseline composed of similar configurations with DANN but without ALs and adaptive input size choice. The first column of Table 3(a) only has one AL layer and the accuracy comparison demonstrates the benefits of *exploring contexts as early as possible*. The right column of Table 3(a) shows the performance gains as the number of AL increases, which verifies the advantages of the inserted recurrent layer. Table 3(b) uses 6AL_VPP_3FC to study the impact of $T$ and the results prove that larger $T$ leads to better performance. This is perhaps due to larger contexts embedded into DANN which are more suitable to capture semantic information.

Table 3: Performance comparison with different configurations of DANN on UCF101 split 1.

(a) Impact of the order and the number of AL using $T = 3$.

| Architecture | Acc. | Architecture | Acc. |
|---|---|---|---|
| B_6VC_3FC | 80.2 | 2AL_4VC_VPP_3FC | 85.9 |
| AL_5VC_VPP_3FC | 85.1 | 3AL_3VC_VPP_3FC | 86.7 |
| VC_AL_4VC_VPP_3FC | 83.3 | 4AL_2VC_VPP_3FC | 86.4 |
| 2VC_AL_3VC_VPP_3FC | 82.4 | 5AL_VC_VPP_3FC | 87.5 |
| 3VC_AL_2VC_VPP_3FC | 82.7 | 6AL_VPP_3FC | 87.9 |
| 4VC_AL_VC_VPP_3FC | 81.4 | | |
| 5VC_AL_VPP_3FC | 80.9 | | |

(b) Impact of $T$.

| Architecture | Acc. |
|---|---|
| 6AL_VPP_3FC, $T = 3$ | 87.9 |
| 6AL_VPP_3FC, $T = 4$ | 88.5 |
| 6AL_VPP_3FC, $T = 5$ | 88.3 |
| 6AL_VPP_3FC, $T = 6$ | 89.0 |

**Combining spatial stream.** Recent work [29] demonstrates that combining appearance information learned from spatial stream can improve the performance of pure 3D CNN. We examine this issue and train a network with static RGB frames similar to [22] by inputting $256 \times 256$ frames and cropping them randomly into $224 \times 224$ regions. The VGG-16 [23] network pre-trained on ImageNet is fine-tuned on UCF101 and HMDB51 separately. Following good practice in [32], we apply weighted averaging of $0.4$ and $0.6$ for RGB and DANN scores, respectively. Table 4 reports the final results of our best model and its fusion with spatial stream on the three splits of both datasets.

**Comparison with the state-of-the-art.** Table 4 reports the best DANN model and state-of-the-art approaches over three splits on UCF101 and HMDB51 datasets in terms of video-level accuracy. As can be seen from Table 4, trajectory-based features are still competitive in the area of deep learning, especially with the help of high-order encodings or deep architectures. Fusion strategies often outperform pure single deep networks. Note that all the other deep networks use a pre-defined temporal length to generate video clip as input such as 16-frame [28] and 60-frame [29], while our DANN determines it in an adaptive manner. Combined with spatial stream, DANN achieves the accuracy of 65.9% and 91.6% on HMDB51 and UCF101, separately.

Table 4: Comparison with the state-of-the-art on HMDB51 and UCF101 (over three splits).

|  | Method | HMDB | UCF |  | Method | HMDB | UCF |
|---|---|---|---|---|---|---|---|
| CNN | Slow fusion [13] | - | 65.4 | Fusion | Two-stream [22] | 59.4 | 88.0 |
|  | C3D [28] | - | 85.2 |  | CNN+deep LSTM [35] | - | 88.6 |
|  | Two-Stream(spatial) [22] | 40.5 | 73.0 |  | TDD [31] | 63.2 | 90.3 |
|  | Two-Stream(temporal) [22] | 54.6 | 83.7 |  | TDD+iDT [31] | 65.9 | 91.5 |
|  | LTC [29] | 57.9 | 83.3 |  | C3D+iDT [28] | - | 90.4 |
|  | Very deep (temporal) [32] | - | 87.0 |  | Very deep (two-stream) [32] | - | 91.4 |
|  | Very deep (spatial) [32] | - | 87.0 |  | LTC+spatial | 61.5 | 88.6 |
| Hand | IDT+FV [30] | 57.2 | 85.9 | Ours | DANN | 63.3 | 89.2 |
|  | IDT+HSV [19] | 61.1 | 87.9 |  | DANN+spatial | 65.9 | **91.6** |
|  | IDT+MIFS [16] | 65.1 | 89.1 |  |  |  |  |
|  | IDT+SFV [20] | **66.8** | - |  |  |  |  |

## 4.3 Qualitative Analysis

We present qualitative analysis of DANN and investigate where mistakes exist. We examine the per-class accuracies are computed and the difference between 6AL_VPP_3FC($T = 6$) and B_6VC_3FC. The class with the largest improvement when 6AL_VPP_3FC($T = 6$) is used instead of B_6VC_3FC is "bowling". This action is composed of preparing for a few seconds and then throwing a bowl. The adaptive temporal choice determined by DANN can aggregate more reasonable semantic structures while B_6VC_3FC has to choose temporal size manually. Figure 6 illustrates sample frames from class "bowling". It is clear that DANN is more likely to leverage reasonable video clips as network input. On the other hand, there are also a few classes that B_6VC_3FC outperforms 6AL_VPP_3FC($T = 6$) such as "haircut". We also illustrate its sample frames in Figure 6. We speculate this phenomenon is partly due to the rich contexts provided by 6AL_VPP_3FC($T = 6$) are not fit to simple actions performed in simple background.

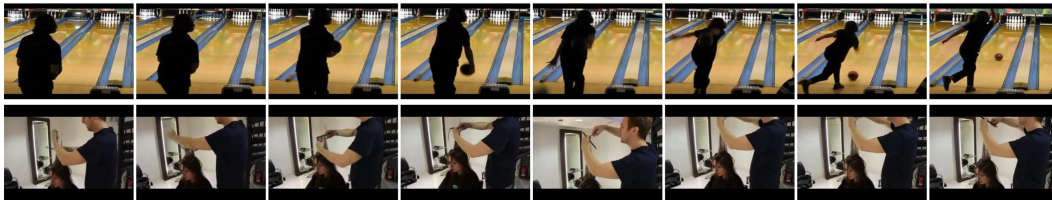

Figure 6: Sample frames of "bowling" and "haircut". For "bowling" 6AL_VPP_3FC($T = 6$) segments video clips with 52 frames (Figure 2) which preserves temporal semantic structures. Such adaptive choice performs worse than baseline for "haircut", where background and action are simple.

## 5 Conclusion and Future Work

This paper introduces a deep alternative neural network (DANN) for action recognition. DANN stacks alternative layers which consists of a volumetric convolutional layer and a recurrent layer. To preserve motion structures in temporal domain, we present an adaptive method to determine the temporal size of network input and develop a volumetric pyramid pooling layer to resize the output before fully connected layers into fixed-size vector. We demonstrate the advantages of DANN on HMDB51 and UCF101 benchmarks and report competitive or superior results to the state-of-the-art.

There still remains some potential area of improvement. The most prominent one is the input size. Although in our model we use adaptive temporal length, the spatial size is still chosen in ad hoc manner. A more compact input such as *action tube* [7] of arbitrary size will be studied in the future, which only contains key actor and spatiotemporal movement regions.

**Acknowledgements.** The work was supported by Shenzhen Peacock Plan (20130408-183003656).

# References

[1] Thomas Brox, Andrés Bruhn, Nils Papenberg, and Joachim Weickert. High accuracy optical flow estimation based on a theory for warping. In *ECCV*, pages 25–36. 2004.

[2] Ronan Collobert, Koray Kavukcuoglu, and Clément Farabet. Torch7: A matlab-like environment for machine learning. In *BigLearn, NIPS Workshop*, number EPFL-CONF-192376, 2011.

[3] Peter Dayan and Laurence F Abbott. *Theoretical neuroscience*, volume 806. Cambridge, MA: MIT Press, 2001.

[4] Gustavo Deco and Tai Sing Lee. The role of early visual cortex in visual integration: a neural model of recurrent interaction. *European Journal of Neuroscience*, 20(4):1089–1100, 2004.

[5] Jeffrey Donahue, Lisa Anne Hendricks, Sergio Guadarrama, Marcus Rohrbach, Subhashini Venugopalan, Kate Saenko, and Trevor Darrell. Long-term recurrent convolutional networks for visual recognition and description. In *ICCV*, pages 2625–2634, 2015.

[6] Ross Girshick, Jeff Donahue, Trevor Darrell, and Jagannath Malik. Rich feature hierarchies for accurate object detection and semantic segmentation. In *CVPR*, pages 580–587, 2014.

[7] Georgia Gkioxari and Jitendra Malik. Finding action tubes. In *CVPR*, pages 759–768, 2015.

[8] Kaiming He, Xiangyu Zhang, Shaoqing Ren, and Jian Sun. Spatial pyramid pooling in deep convolutional networks for visual recognition. *TPAMI*, 37(9):1904–1916, 2015.

[9] David H Hubel and Torsten N Wiesel. Receptive fields of single neurones in the cat's striate cortex. *The Journal of physiology*, 148(3):574–591, 1959.

[10] David H Hubel and Torsten N Wiesel. Receptive fields, binocular interaction and functional architecture in the cat's visual cortex. *The Journal of physiology*, 160(1):106–154, 1962.

[11] Shuiwang Ji, Wei Xu, Ming Yang, and Kai Yu. 3d convolutional neural networks for human action recognition. *TPAMI*, 35(1):221–231, 2013.

[12] Vadim Kantorov and Ivan Laptev. Efficient feature extraction, encoding and classification for action recognition. In *CVPR*, pages 2593–2600, 2014.

[13] Andrej Karpathy, George Toderici, Sachin Shetty, Tommy Leung, Rahul Sukthankar, and Li Fei-Fei. Large-scale video classification with convolutional neural networks. In *CVPR*, pages 1725–1732, 2014.

[14] Alex Krizhevsky, Ilya Sutskever, and Geoffrey E Hinton. Imagenet classification with deep convolutional neural networks. In *NIPS*, pages 1097–1105, 2012.

[15] Hildegard Kuehne, Hueihan Jhuang, Estíbaliz Garrote, Tomaso Poggio, and Thomas Serre. Hmdb: a large video database for human motion recognition. In *ICCV*, pages 2556–2563, 2011.

[16] Zhengzhong Lan, Ming Lin, Xuanchong Li, Alex G Hauptmann, and Bhiksha Raj. Beyond gaussian pyramid: Multi-skip feature stacking for action recognition. In *CVPR*, pages 204–212, 2015.

[17] Ming Liang and Xiaolin Hu. Recurrent convolutional neural network for object recognition. In *CVPR*, pages 3367–3375, 2015.

[18] Ming Liang, Xiaolin Hu, and Bo Zhang. Convolutional neural networks with intra-layer recurrent connections for scene labeling. In *NIPS*, pages 937–945, 2015.

[19] Xiaojiang Peng, Limin Wang, Xingxing Wang, and Yu Qiao. Bag of visual words and fusion methods for action recognition: Comprehensive study and good practice. *arXiv preprint arXiv:1405.4506*, 2014.

[20] Xiaojiang Peng, Changqing Zou, Yu Qiao, and Qiang Peng. Action recognition with stacked fisher vectors. In *ECCV*, pages 581–595. 2014.

[21] Pedro Pinheiro and Ronan Collobert. Recurrent convolutional neural networks for scene labeling. In *ICML*, pages 82–90, 2014.

[22] Karen Simonyan and Andrew Zisserman. Two-stream convolutional networks for action recognition in videos. In *NIPS*, pages 568–576, 2014.

[23] Karen Simonyan and Andrew Zisserman. Very deep convolutional networks for large-scale image recognition. *arXiv preprint arXiv:1409.1556*, 2014.

[24] Khurram Soomro, Amir Roshan Zamir, and Mubarak Shah. Ucf101: A dataset of 101 human actions classes from videos in the wild. *arXiv preprint arXiv:1212.0402*, 2012.

[25] Nitish Srivastava, Elman Mansimov, and Ruslan Salakhudinov. Unsupervised learning of video representations using lstms. In *ICML*, pages 843–852, 2015.

[26] Christian Szegedy, Wei Liu, Yangqing Jia, Pierre Sermanet, Scott Reed, Dragomir Anguelov, Dumitru Erhan, Vincent Vanhoucke, and Andrew Rabinovich. Going deeper with convolutions. In *CVPR*, pages 1–9, 2015.

[27] Graham W Taylor, Rob Fergus, Yann LeCun, and Christoph Bregler. Convolutional learning of spatio-temporal features. In *ECCV*, pages 140–153. 2010.

[28] Du Tran, Lubomir Bourdev, Rob Fergus, Lorenzo Torresani, and Manohar Paluri. Learning spatiotemporal features with 3d convolutional networks. In *ICCV*, pages 4489–4497, 2015.

[29] Gül Varol, Ivan Laptev, and Cordelia Schmid. Long-term temporal convolutions for action recognition. *arXiv preprint arXiv:1604.04494*, 2016.

[30] Heng Wang and Cordelia Schmid. Action recognition with improved trajectories. In *ICCV*, pages 3551–3558, 2013.

[31] Limin Wang, Yu Qiao, and Xiaoou Tang. Action recognition with trajectory-pooled deep-convolutional descriptors. In *CVPR*, pages 4305–4314, 2015.

[32] Limin Wang, Yuanjun Xiong, Zhe Wang, and Yu Qiao. Towards good practices for very deep two-stream convnets. *arXiv preprint arXiv:1507.02159*, 2015.

[33] RL Waters and JM Morris. Electrical activity of muscles of the trunk during walking. *Journal of anatomy*, 111(Pt 2):191, 1972.

[34] Paul J Werbos. Backpropagation through time: what it does and how to do it. *Proceedings of the IEEE*, 78(10):1550–1560, 1990.

[35] Joe Yue-Hei Ng, Matthew Hausknecht, Sudheendra Vijayanarasimhan, Oriol Vinyals, Rajat Monga, and George Toderici. Beyond short snippets: Deep networks for video classification. In *CVPR*, pages 4694–4702, 2015.

[36] Christopher Zach, Thomas Pock, and Horst Bischof. A duality based approach for realtime tv-l 1 optical flow. In *Pattern Recognition*, pages 214–223. 2007.

[37] Matthew D Zeiler and Rob Fergus. Stochastic pooling for regularization of deep convolutional neural networks. *arXiv preprint arXiv:1301.3557*, 2013.

